# Approximate Planning in Large POMDPs via Reusable Trajectories

**Michael Kearns**
AT&T Labs
mkearns@research.att.com

**Yishay Mansour**
Tel Aviv University
mansour@math.tau.ac.il

**Andrew Y. Ng**
UC Berkeley
ang@cs.berkeley.edu

## Abstract

We consider the problem of reliably choosing a near-best strategy from a restricted class of strategies $\Pi$ in a partially observable Markov decision process (POMDP). We assume we are given the ability to *simulate* the POMDP, and study what might be called the *sample complexity* — that is, the amount of data one must generate in the POMDP in order to choose a good strategy. We prove upper bounds on the sample complexity showing that, even for *infinitely large and arbitrarily complex* POMDPs, the amount of data needed can be finite, and depends only linearly on the complexity of the restricted strategy class $\Pi$, and exponentially on the horizon time. This latter dependence can be eased in a variety of ways, including the application of gradient and local search algorithms. Our measure of complexity generalizes the classical supervised learning notion of VC dimension to the settings of reinforcement learning and planning.

## 1 Introduction

Much recent attention has been focused on partially observable Markov decision processes (POMDPs) which have exponentially or even infinitely large state spaces. For such domains, a number of interesting basic issues arise. As the state space becomes large, the classical way of specifying a POMDP by tables of transition probabilities clearly becomes infeasible. To intelligently discuss the problem of planning — that is, computing a good strategy [1] in a given POMDP — *compact* or implicit representations of both POMDPs, and of strategies in POMDPs, must be developed. Examples include factored next-state distributions [2, 3, 7], and strategies derived from function approximation schemes [8]. The trend towards such compact representations, as well as algorithms for planning and learning using them, is reminiscent of supervised learning, where researchers have long emphasized parametric models (such as decision trees and neural networks) that can capture only limited structure, but which enjoy a number of computational and information-theoretic benefits.

Motivated by these issues, we consider a setting were we are given a *generative model*, or

simulator, for a POMDP, and wish to find a good strategy $\pi$ from some restricted class of strategies $\Pi$. A generative model is a "black box" that allows us to generate experience (trajectories) from different states of our choosing. Generative models are an abstract notion of compact POMDP representations, in the sense that the compact representations typically considered (such as factored next-state distributions) already provide efficient generative models. Here we are imagining that the strategy class $\Pi$ is given by some compact representation or by some natural limitation on strategies (such as bounded memory). Thus, the view we are adopting is that even though the world (POMDP) may be extremely complex, we assume that we can at least simulate or sample experience in the world (via the generative model), and we try to use this experience to choose a strategy from some "simple" class $\Pi$.

We study the following question: How many calls to a generative model are needed to have enough data to choose a near-best strategy in the given class? This is analogous to the question of *sample complexity* in supervised learning — but harder. The added difficulty lies in the *reuse* of data. In supervised learning, *every* sample $\langle x, f(x) \rangle$ provides feedback about *every* hypothesis function $h(x)$ (namely, how close $h(x)$ is to $f(x)$). If $h$ is restricted to lie in some hypothesis class $\mathcal{H}$, this reuse permits sample complexity bounds that are far smaller than the size of $\mathcal{H}$. For instance, only $O(\log(|\mathcal{H}|))$ samples are needed to choose a near-best model from a finite class $\mathcal{H}$. If $\mathcal{H}$ is infinite, then sample sizes are obtained that depend only on some measure of the *complexity* of $\mathcal{H}$ (such as VC dimension [9]), but which have *no dependence* on the complexity of the target function or the size of the input domain.

In the POMDP setting, we would like analogous sample complexity bounds in terms of the "complexity" of the strategy class $\Pi$ — bounds that have no dependence on the size or complexity of the POMDP. But unlike the supervised learning setting, experience "reuse" is not immediate in POMDPs. To see this, consider the "straw man" algorithm that, starting with some $\pi \in \Pi$, uses the generative model to generate many trajectories under $\pi$, and thus forms a Monte Carlo estimate of $V^\pi(s_0)$. It is not clear that these trajectories under $\pi$ are of much use in evaluating a different $\pi' \in \Pi$, since $\pi$ and $\pi'$ may quickly disagree on which actions to take. The naive Monte Carlo method thus gives $O(|\Pi|)$ bounds on the "sample complexity," rather than $O(\log(|\Pi|))$, for the finite case.

In this paper, we shall describe the *trajectory tree* method of generating "reusable" trajectories, which requires generating only a (relatively) small number of trajectories — a number that is independent of the state-space size of the POMDP, depends only linearly on a general measure of the *complexity* of the strategy class $\Pi$, and depends exponentially on the horizon time. This latter dependence can be eased via gradient algorithms such as Williams' REINFORCE [10] and Baird and Moore's more recent VAPS [1], and by local search techniques. Our measure of strategy class complexity generalizes the notion of VC dimension in supervised learning to the settings of reinforcement learning and planning, and we give bounds that recover for these settings the most powerful analogous results in supervised learning — bounds for arbitrary, infinite strategy classes that depend only on the dimension of the class rather than the size of the state space.

## 2   Preliminaries

We begin with some standard definitions. A **Markov decision process (MDP)** is a tuple $(S, s_0, A, \{P(\cdot|s,a)\}, R)$, where: $S$ is a (possibly infinite) **state set**; $s_0 \in S$ is a **start state**; $A = \{a_1, \ldots, a_k\}$ are **actions**; $P(\cdot|s,a)$ gives the next-state distribution upon taking action $a$ from state $s$; and the reward function $R(s,a)$ gives the corresponding rewards. We assume for simplicity that rewards are deterministic, and further that they are bounded

in absolute value by $R_{\max}$. A **partially observable Markov decision process (POMDP)** consists of an underlying MDP and **observation distributions** $Q(o|s)$ for each state $s$, where $o$ is the random **observation** made at $s$.

We have adopted the common assumption of a fixed start state,[2] because once we limit the class of strategies we entertain, there may not be a single "best" strategy in the class—different start states may have different best strategies in $\Pi$. We also assume that we are given a POMDP $M$ in the form of a **generative model** for $M$ that, when given as input any state-action pair $(s, a)$, will output a state $s'$ drawn according to $P(\cdot|s, a)$, an observation $o$ drawn according to $Q(\cdot|s)$, and the reward $R(s, a)$. This gives us the ability to *sample* the POMDP $M$ in a random-access way. This definition may initially seem unreasonably generous: the generative model is giving us a fully observable simulation of a partially observable process. However, the key point is that we must still find a strategy that performs well in the *partially observable* setting. As a concrete example, in designing an elevator control system, we may have access to a simulator that generates random rider arrival times, and keeps track of the waiting time of each rider, the number of riders waiting at every floor at every time of day, and so on. However helpful this information might be in *designing* the controller, this controller must only *use* information about which floors currently have had their call button pushed (the observables). In any case, readers uncomfortable with the power provided by our generative models are referred to Section 5, where we briefly describe results requiring only an extremely weak form of partially observable simulation.

At any time $t$, the agent will have seen some sequence of observations, $o_0, \ldots, o_t$, and will have chosen actions and received rewards for each of the $t$ time steps prior to the current one. We write its **observable history** as $h = \langle (o_0, a_0, r_0), \ldots, (o_{t-1}, a_{t-1}, r_{t-1}), (o_t, \_, \_) \rangle$. Such observable histories, also called **trajectories**, are the inputs to strategies. More formally, a **strategy** $\pi$ is any (stochastic) mapping from observable histories to actions. (For example, this includes approaches which use the observable history to track the *belief state* [5].) A **strategy class** $\Pi$ is any set of strategies.

We will restrict our attention to the case of discounted return,[3] and we let $\gamma \in [0, 1)$ be the discount factor. We define the $\epsilon$-**horizon time** to be $H_\epsilon = \log_\gamma(\epsilon(1 - \gamma)/2R_{\max})$. Note that returns beyond the first $H_\epsilon$-steps can contribute at most $\epsilon/2$ to the total discounted return. Also, let $V_{\max} = R_{\max}/(1 - \gamma)$ bound the value function. Finally, for a POMDP $M$ and a strategy class $\Pi$, we define $opt(M, \Pi) = \sup_{\pi \in \Pi} V^\pi(s_0)$ to be the best expected return achievable from $s_0$ using $\Pi$.

Our problem is thus the following: Given a generative model for a POMDP $M$ and a strategy class $\Pi$, how many calls to the generative model must we make, in order to have enough data to choose a $\pi \in \Pi$ whose performance $V^\pi(s_0)$ approaches $opt(M, \Pi)$? Also, *which* calls should we make to the generative model to achieve this?

## 3  The Trajectory Tree Method

We now describe how we can use a generative model to create "reusable" trajectories. For ease of exposition, we assume there are only two actions $a_1$ and $a_2$, but our results generalize easily to any finite number of actions. (See the full paper [6].)

A *trajectory tree* is a binary tree in which each node is labeled by a state and observation pair, and has a child for each of the two actions. Additionally, each link to a child is labeled by a reward, and the tree's depth will be $H_\epsilon$, so it will have about $2^{H_\epsilon}$ nodes. (In Section 4, we will discuss settings where this exponential dependence on $H_\epsilon$ can be eased.) Each trajectory tree is built as follows: The root is labeled by $s_0$ and the observation there, $o_0$. Its two children are then created by calling the generative model on $(s_0, a_1)$ and $(s_0, a_2)$, which gives us the two next-states reached (say $s_1'$ and $s_2'$ respectively), the two observations made (say $o_1'$ and $o_2'$), and the two rewards received ($r_1' = R(s_0, a_1)$ and $r_2' = R(s_0, a_2)$). Then $(s_1', o_1')$ and $(s_2', o_2')$ label the root's $a_1$-child and $a_2$-child, and the links to these children are labeled $r_1'$ and $r_2'$. Recursively, we generate two children and rewards this way for each node down to depth $H_\epsilon$.

Now for any *deterministic* strategy $\pi$ and any trajectory tree $T$, $\pi$ defines a path through $T$: $\pi$ starts at the root, and inductively, if $\pi$ is at some internal node in $T$, then we feed to $\pi$ the observable history along the path from the root to that node, and $\pi$ selects and moves to a child of the current node. This continues until a leaf node is reached, and we define $R(\pi, T)$ to be the discounted sum of returns along the path taken. In the case that $\pi$ is stochastic, $\pi$ defines a *distribution* on paths in $T$, and $R(\pi, T)$ is the expected return according to this distribution. (We will later also describe another method for treating stochastic strategies.) Hence, given $m$ trajectory trees $T_1, \ldots, T_m$, a natural estimate for $V^\pi(s_0)$ is $\hat{V}^\pi(s_0) = \frac{1}{m} \sum_{i=1}^{m} R(\pi, T_i)$. Note that each tree can be used to evaluate *any* strategy, much the way a single labeled example $\langle x, f(x) \rangle$ can be used to evaluate any hypothesis $h(x)$ in supervised learning. Thus in this sense, trajectory trees are *reusable*.

Our goal now is to establish *uniform convergence* results that bound the error of the estimates $\hat{V}^\pi(s_0)$ as a function of the "sample size" (number of trees) $m$. Section 3.1 first treats the easier case of deterministic classes $\Pi$; Section 3.2 extends the result to stochastic classes.

## 3.1   The Case of Deterministic $\Pi$

Let us begin by stating a result for the special case of finite classes of deterministic strategies, which will serve to demonstrate the kind of bound we seek.

**Theorem 3.1** *Let $\Pi$ be any finite class of deterministic strategies for an arbitrary two-action POMDP $M$. Let $m$ trajectory trees be created using a generative model for $M$, and $\hat{V}^\pi(s_0)$ be the resulting estimates. If $m = O\left((V_{\max}/\epsilon)^2(\log(|\Pi|) + \log(1/\delta))\right)$, then with probability $1 - \delta$, $|V^\pi(s_0) - \hat{V}^\pi(s_0)| \leq \epsilon$ holds simultaneously for all $\pi \in \Pi$.*

Due to space limitations, detailed proofs of the results of this section are left to the full paper [6], but we will try to convey the intuition behind the ideas. Observe that for any *fixed* deterministic $\pi$, the estimates $R(\pi, T_i)$ that are generated by the $m$ different trajectory trees $T_i$ are independent. Moreover, each $R(\pi, T_i)$ is an unbiased estimate of the expected discounted $H_\epsilon$-step return of $\pi$, which is in turn $\epsilon/2$-close to $V^\pi(s_0)$. These observations, combined with a simple Chernoff and union bound argument, are sufficient to establish Theorem 3.1. Rather than developing this argument here, we instead move straight on to the harder case of infinite $\Pi$.

When addressing sample complexity in supervised learning, perhaps the most important insight is that even though a class $\mathcal{H}$ may be infinite, the number of possible *behaviors* of $\mathcal{H}$ on a finite set of points is often not exhaustive. More precisely, for boolean functions, we say that the set $x_1, \ldots, x_d$ is *shattered* by $\mathcal{H}$ if every of the $2^d$ possible labelings of

these points is realized by some $h \in \mathcal{H}$. The VC dimension of $\mathcal{H}$ is then defined as the size of the largest shattered set [9]. It is known that if the VC dimension of $\mathcal{H}$ is $d$, then the number $\Phi_d(m)$ of possible labelings induced by $\mathcal{H}$ on a set of $m$ points is at most $(em/d)^d$, which is much less than $2^m$ for $d \ll m$. This fact provides the key leverage exploited by the classical VC dimension results, and we will concentrate on replicating this leverage in our setting.

If $\Pi$ is a (possibly infinite) set of deterministic strategies, then each strategy $\pi \in \Pi$ is simply a deterministic function mapping from the set of observable histories to the set $\{a_1, a_2\}$, and is thus a boolean function on observable histories. We can therefore write $\text{VC}(\Pi)$ to denote the familiar VC dimension of the set of binary functions $\Pi$. For example, if $\Pi$ is the set of all thresholded linear functions of the current vector of observations (a particular type of memoryless strategy), then $\text{VC}(\Pi)$ simply equals the number of parameters. We now show intuitively why a class $\Pi$ of bounded VC dimension $d$ cannot induce exhaustive behavior on a set $T_1, \ldots, T_m$ of trajectory trees for $m \gg d$. Note that if $\pi_1, \pi_2 \in \Pi$ are such that their "reward labelings" $\langle R(\pi_1, T_1), \ldots, R(\pi_1, T_m) \rangle$ and $\langle R(\pi_2, T_1), \ldots, R(\pi_2, T_m) \rangle$ differ, then $R(\pi_1, T_i) \neq R(\pi_2, T_i)$ for some $1 \leq i \leq m$. But if $\pi_1$ and $\pi_2$ give different returns on $T_i$, then they must choose different actions at some node in $T_i$. In other words, every different reward labeling of the set of $m$ *trees* yields a different (binary) labeling of the set of $m \cdot 2^{H_\epsilon}$ observable *histories* in the trees. So, the number of different tree reward labelings can be at most $\Phi_d(m \cdot 2^{H_\epsilon}) \leq (em \cdot 2^{H_\epsilon}/d)^d$. By developing this argument carefully and applying classical uniform convergence techniques, we obtain the following theorem. (Full proof in [6].)

**Theorem 3.2** *Let* $\Pi$ *be any class of deterministic strategies for an arbitrary two-action POMDP* $M$, *and let* $\text{VC}(\Pi)$ *denote its VC dimension. Let* $m$ *trajectory trees be created using a generative model for* $M$, *and* $\hat{V}^\pi(s_0)$ *be the resulting estimates. If*

$$m = O\left((V_{\max}/\epsilon)^2 (H_\epsilon \text{VC}(\Pi) \log(V_{\max}/\epsilon) + \log(1/\delta))\right) \tag{1}$$

*then with probability* $1 - \delta$, $|V^\pi(s_0) - \hat{V}^\pi(s_0)| \leq \epsilon$ *holds simultaneously for all* $\pi \in \Pi$.

## 3.2 The Case of Stochastic $\Pi$

We now address the case of stochastic strategy classes. We describe an approach where we *transform* stochastic strategies into "equivalent" deterministic ones and operate on the deterministic versions, reducing the problem to the one handled in the previous section. The transformation is as follows: Given a class of stochastic strategies $\Pi$, each with domain $X$ (where $X$ is the set of all observable histories), we first extend the domain to be $X \times [0, 1]$. Now for each stochastic strategy $\pi \in \Pi$, define a corresponding deterministic *transformed* strategy $\pi'$ with domain $X \times [0, 1]$, given by: $\pi'(h, r) = a_1$ if $r \leq \mathbf{Pr}[\pi(h) = a_1]$, and $\pi'(h, r) = a_2$ otherwise (for any $h \in X$, $r \in [0, 1]$). Let $\Pi'$ be the collection of these transformed deterministic strategies $\pi'$. Since $\Pi'$ is just a set of deterministic boolean functions, its VC dimension is well-defined. We then define the *pseudo-dimension* of the *original* set of stochastic strategies $\Pi$ to be $\text{pVC}(\Pi) = \text{VC}(\Pi')$.[4]

Having transformed the strategy class, we also need to transform the POMDP, by augmenting the state space $S$ to be $S \times [0, 1]$. Informally, the transitions and rewards remain the same, except that after each state transition, we draw a new random variable $r$ uniformly in $[0, 1]$, and independently of all previous events. States are now of the form $(s, r)$, and we let $r$ be an observed variable. Whenever in the original POMDP a stochastic strategy $\pi$ would

have been given a history $h$, in the transformed POMDP the corresponding deterministic transformed strategy $\pi'$ is given $(h, r)$, where $r$ is the $[0, 1]$-random variable at the current state. By the definition of $\pi'$, it is easy to see that $\pi'$ and $\pi$ have exactly the same chance of choosing each action at any node (randomization over $r$).

We are now back in the deterministic case, so Theorem 3.2 applies, with VC($\Pi$) replaced by pVC($\Pi$) = VC($\Pi'$), and we again have the desired uniform convergence result.

## 4 Algorithms for Approximate Planning

Given a generative model for a POMDP, the preceding section's results immediately suggest a class of approximate planning algorithms: generate $m$ trajectory trees $T_1, \ldots, T_m$, and search for a $\pi \in \Pi$ that maximizes $\hat{V}^\pi(s_0) = (1/m) \sum R(\pi, T_i)$. The following corollary to the uniform convergence results establishes the soundness of this approach.

**Corollary 4.1** *Let $\Pi$ be a class of strategies in a POMDP $M$, and let the number $m$ of trajectory trees be as given in Theorem 3.2. Let $\hat{\pi} = \arg\max_{\pi \in \Pi} \{\hat{V}^\pi(s_0)\}$ be the policy in $\Pi$ with the highest empirical return on the $m$ trees. Then with probability $1 - \delta$, $\hat{\pi}$ is near-optimal within $\Pi$:*

$$V^{\hat{\pi}}(s_0) \geq opt(M, \Pi) - 2\epsilon. \qquad (2)$$

If the suggested maximization is computationally infeasible, one can search for a local maximum $\pi$ instead, and uniform convergence again assures us that $\hat{V}^\pi(s_0)$ is a trusted estimate of our true performance. Of course, even finding a local maximum can be expensive, since each trajectory tree is of size exponential in $H_\epsilon$.

However, in practice it may be possible to significantly reduce the cost of the search. Suppose we are using a class of (possibly transformed) deterministic strategies, and we perform a greedy local search over $\Pi$ to optimize $\hat{V}^\pi(s_0)$. Then at any time in the search, to evaluate the policy we are currently considering, we really need to look at only a single path of length $H_\epsilon$ in each tree, corresponding to the path taken by the strategy being considered. Thus, we should build the trajectory trees *lazily* — that is, incrementally build each node of each tree only as it is needed to evaluate $R(\pi, T_i)$ for the current strategy $\pi$. If there are parts of a tree that are reached only by poor policies, then a good search algorithm may never even build these parts of the tree. In any case, for a fixed number of trees, each step of the local search now takes time only *linear* in $H_\epsilon$.[5]

There is a different approach that works directly on stochastic strategies (that is, without requiring the transformation to deterministic strategies). In this case each stochastic strategy $\pi$ defines a distribution over *all* the paths in a trajectory tree, and thus calculating $R(\pi, T)$ may in general require examining complete trees. However, we can view each trajectory tree as a small, deterministic POMDP by itself, with the children of each node in the tree being its successor nodes. So if $\Pi = \{\pi_\theta : \theta \in \mathbb{R}^d\}$ is a smoothly parameterized family of stochastic strategies, then algorithms such as William's REINFORCE [10] can be used to find an unbiased estimate of the gradient $(d/d\theta)\hat{V}^{\pi_\theta}(s_0)$, which in turn can be used to

perform stochastic gradient ascent to maximize $\hat{V}^{\pi_\theta}(s_0)$. Moreover, for a fixed number of trees, these algorithms need only $O(H_\epsilon)$ time per gradient estimate; so combined with lazy tree construction, we again have a practical algorithm whose per-step complexity is only *linear* in the horizon time. This line of thought is further developed in the long version of the paper.[6]

## 5   The Random Trajectory Method

Using a fully observable generative model of a POMDP, we have shown that the trajectory tree method gives uniformly good value estimates, with an amount of experience linear in $VC(\Pi)$, and exponential in $H_\epsilon$. It turns out we can significantly weaken the generative model, yet still obtain essentially the same theoretical results. In this harder case, we assume a generative model that provides only *partially observable* histories generated by a *truly random* strategy (which takes each action with equal probability at every step, regardless of the history so far). Furthermore, these trajectories always begin at the designated start state, so there is no ability provided to "reset" the POMDP to any state other than $s_0$. (Indeed, underlying states may never be observed.)

Our method for this harder case is called the Random Trajectory method. It seems to lead less readily to practical algorithms than the trajectory tree method, and its formal description and analysis, which is more difficult than for trajectory trees, are given in the long version of this paper [6]. As in Theorem 3.2, we prove that the amount of data needed is linear in $VC(\Pi)$, and exponential in the horizon time — that is, by averaging appropriately over the resulting ensemble of trajectories generated, this amount of data is sufficient to yield uniformly good estimates of the values for all strategies in $\Pi$.

## Footnotes

[1] Throughout, we use the word *strategy* to mean any mapping from observable histories to actions, which generalizes the notion of policy in a fully observable MDP.

[2] An equivalent definition is to assume a fixed distribution $D$ over start states, since $s_0$ can be a "dummy" state whose next-state distribution under any action is $D$.

[3] The results in this paper can be extended without difficulty to the undiscounted finite-horizon setting [6].

[4]This is equivalent to the conventional definition of the pseudo-dimension of $\Pi$ [4], when it is viewed as a set of maps into real-valued action-probabilities.

[5] See also (Ng and Jordan, in preparation) which, by assuming a much stronger model of a POMDP (a deterministic function $f$ such that $f(s, a, r)$ is distributed according to $P(\cdot|s, a)$ when $r$ is distributed Uniform[0,1]), gives an algorithm that enjoys uniform convergence bounds similar to those presented here, but with only a polynomial rather than exponential dependence on $H_\epsilon$. The algorithm samples a number of vectors $r^{(i)} \in [0, 1]^{H_\epsilon}$, each of which, with $f$, defines an $H_\epsilon$-step Monte Carlo evaluation trial for any policy $\pi$. The bound is on the number of such random vectors needed (rather than on the total number of calls to $f$).

[6]In the full paper, we also show how these algorithms can be extended to find in expected $O(H_\epsilon)$ time an unbiased estimate of the gradient of the *true* value $V^{\pi_\theta}(s_0)$ for discounted infinite horizon problems (whereas most current algorithms either only converge asymptotically to an unbiased estimate of this gradient, or need an absorbing state and "proper" strategies).

## References

[1] L. Baird and A. W. Moore. Gradient descent for general Reinforcement Learning. In *Advances in Neural Information Processing Systems 11*, 1999.

[2] C. Boutilier, T. Dean, and S. Hanks. Decision theoretic planning: Structural assumptions and computational leverage. *Journal of Artificial Intelligence Research*, 1999.

[3] X. Boyen and D. Koller. Tractable inference for complex stochastic processes. In *Proc. UAI*, pages 33–42, 1998.

[4] David Haussler. Decision theoretic generalizations of the PAC model for neural net and oter learning applications. *Information and Computation*, 100:78–150, 1992.

[5] L. P. Kaelbling, M. L. Littman, and A. R. Cassandra. Planning and acting in partially observable stochastic domains. *Artificial Intelligence*, 101, 1998.

[6] M. Kearns, Y. Mansour, and A. Y. Ng. Approximate planning in large POMDPs via reusable trajectories. (long version), 1999.

[7] D. Koller and R. Parr. Computing factored value functions for policies in structured MDPs. In *Proceedings of the Sixteenth International Joint Conference on Artificial Intelligence*, 1999.

[8] R. S. Sutton and A. G. Barto. *Reinforcement Learning*. MIT Press, 1998.

[9] V.N. Vapnik. *Estimation of Dependences Based on Empirical Data*. Springer-Verlag, 1982.

[10] R. J. Williams. Simple statistical gradient-following algorithms for connectionist reinforcement learning. *Machine Learning*, 8:229–256, 1992.

